# Mandatory Leaf Node Prediction in Hierarchical Multilabel Classification

**Wei Bi**     **James T. Kwok**
Department of Computer Science and Engineering
Hong Kong University of Science and Technology
Clear Water Bay, Hong Kong
{weibi,jamesk}@cse.ust.hk

## Abstract

In hierarchical classification, the prediction paths may be required to always end at leaf nodes. This is called mandatory leaf node prediction (MLNP) and is particularly useful when the leaf nodes have much stronger semantic meaning than the internal nodes. However, while there have been a lot of MLNP methods in hierarchical multiclass classification, performing MLNP in hierarchical multilabel classification is much more difficult. In this paper, we propose a novel MLNP algorithm that (i) considers the global hierarchy structure; and (ii) can be used on hierarchies of both trees and DAGs. We show that one can efficiently maximize the joint posterior probability of all the node labels by a simple greedy algorithm. Moreover, this can be further extended to the minimization of the expected symmetric loss. Experiments are performed on a number of real-world data sets with tree- and DAG-structured label hierarchies. The proposed method consistently outperforms other hierarchical and flat multilabel classification methods.

## 1 Introduction

In many real-world classification problems, the output labels are organized in a hierarchy. For example, gene functions are arranged in a tree in the Functional Catalog (FunCat) or as a directed acyclic graph (DAG) in the Gene Ontology (GO) [1]; musical signals are organized in an audio taxonomy [2]; and documents in the Wikipedia hierarchy. Hierarchical classification algorithms, which utilize these hierarchical relationships between labels in making predictions, often lead to better performance than traditional non-hierarchical (flat) approaches.

In hierarchical classification, the labels associated with each pattern can be on a path from the root to a leaf (full-path prediction); or stop at an internal node (partial-path prediction [3]). Following the terminology in the recent survey [4], when only full-path predictions are allowed, it is called mandatory leaf node prediction (MLNP); whereas when partial-path predictions are also allowed, it is called non-mandatory leaf node prediction (NMLNP). Depending on the application and how the label hierarchy is generated, either one of these prediction modes may be more relevant. For example, in the taxonomies of musical signals [2] and genes [5], the leaf nodes have much stronger semantic/biological meanings than the internal nodes, and MLNP is more important. Besides, sometimes the label hierarchy is learned from the data, using methods like hierarchical clustering [6], Bayesian network structure learning [7] and label tree methods [8, 9]. In these cases, the internal nodes are only artificial, and MLNP is again more relevant. In the recent Second Pascal Challenge on Large-scale Hierarchical Text Classification, the tasks also require MLNP.

In this paper, we focus on hierarchical multilabel classification (HMC), which differs from hierarchical multiclass classification in that the labels of each pattern can fall on a union of paths in the hierarchy [10]. An everyday example is that a document/image/song/video may have multiple tags. Because of its practical significance, HMC has been extensively studied in recent years [1, 3, 10–12].

While there have been a lot of MLNP methods in hierarchical multiclass classification [4], none of these can be easily extended for the more difficult HMC setting. They all rely on training a multiclass classifier at each node, and then use a recursive strategy to predict which subtree to pursue at the next lower level. In hierarchical multiclass classification, exactly one subtree is to be pursued; whereas in HMC, one has to decide at each node how many and which subtrees to pursue. Even when this can be performed (e.g., by adjusting the classification threshold heuristically), it is difficult to ensure that all the prediction paths will end at leaf nodes, and so a lot of partial paths may be resulted.

Alternatively, one may perform MLNP by first predicting the number of leaf labels ($k$) that the test pattern has, and then pick the $k$ leaf labels whose posterior probabilities are the largest. Prediction of $k$ can be achieved by using the MetaLabeler [13], though this involves another, possibly non-trivial, learning task. Moreover, the posterior probability computed at each leaf $l$ corresponds to a single prediction path from the root to $l$. However, the target multilabel in HMC can have multiple paths. Hence, a better approach is to compute the posterior probabilities of all subtrees/subgraphs that have $k$ leaf nodes; and then pick the one with the largest probability. However, as there are $\binom{N}{k}$ such possible subsets (where $N$ is the number of leafs), this can be expensive when $N$ is large.

Recently, Cerri *et al.* [14] proposed the HMC-label-powerset (HMC-LP), which is specially designed for MLNP in HMC. Its main idea is to reduce the hierarchical problem to a non-hierarchical problem by running the (non-hierarchical) multilabel classification method of label-powerset [15] at each level of the hierarchy. However, this significantly increases the number of "meta-labels", making it unsuitable for large hierarchies. Moreover, as it processes the hierarchy level-by-level, this cannot be applied on DAGs, where "levels" are not well-defined.

In this paper, we propose an efficient algorithm for MLNP in both tree-structured and DAG-structured hierarchical multilabel classification. The target multilabel is obtained by maximizing the posterior probability among all feasible multilabels. By adopting a weak "nested approximation" assumption, we show that the resultant optimization problem can be efficiently solved by a greedy algorithm. Empirical results also demonstrate that this "nested approximation" assumption holds in general. The rest of this paper is organized as follows. Section 2 describes the proposed framework for MLNP on tree-structured hierarchies, which is then extended to DAG-structured hierarchies in Section 3. Experimental results are presented in Section 4, and the last section gives some concluding remarks.

## 2  Maximum a Posteriori MLNP on Label Trees

In this section, we assume that the label hierarchy is a tree $\mathcal{T}$. With a slight abuse of notation, we will also use $\mathcal{T}$ to denote the set of all the tree nodes, which are indexed from 0 (for the root), $1, 2, \ldots, N$. Let the set of leaf nodes in $\mathcal{T}$ be $\mathcal{L}$. For a subset $A \subseteq \mathcal{T}$, its complement is denoted by $A^c = \mathcal{T} \backslash A$. For a node $i$, denote its parent by $\mathrm{pa}(i)$, and its set of children by $\mathrm{child}(i)$. Moreover, given a vector $\mathbf{y}$, $\mathbf{y}_A$ is the subvector of $\mathbf{y}$ with indices from $A$.

In HMC, we are given a set of training examples $\{(\mathbf{x}, \mathbf{y})\}$, where $\mathbf{x}$ is the input and $\mathbf{y} = [y_0, \ldots, y_N]' \in \{0, 1\}^{N+1}$ is the multilabel denoting memberships of $\mathbf{x}$ to each of the nodes. Equivalently, $\mathbf{y}$ can be represented by a set $\Omega \subseteq \mathcal{T}$, such that $y_i = 1$ if $i \in \Omega$; and 0 otherwise. For $\mathbf{y}$ (or $\Omega$) to respect the tree structure, we require that $y_i = 1 \Rightarrow y_{\mathrm{pa}(i)} = 1$ for any non-root node $i \in \mathcal{T}$.

In this paper, we assume that for any group of siblings $\{i_1, i_2, \ldots, i_m\}$, their labels are conditionally independent given the label of their parent $\mathrm{pa}(i_1)$ and $\mathbf{x}$, i.e., $p(y_{i_1}, y_{i_2}, \ldots y_{i_m} | y_{\mathrm{pa}(i_1)}, \mathbf{x}) = \prod_{j=1}^{m} p(y_{i_j} | y_{\mathrm{pa}(i_1)}, \mathbf{x})$. This simplification is standard in Bayesian networks and also commonly used in HMC [16, 17]. By repeated application of the probability product rule, we have

$$p(y_0, \ldots, y_N | \mathbf{x}) = p(y_0 | \mathbf{x}) \prod_{i \in \mathcal{T} \backslash \{0\}} p(y_i \mid y_{\mathrm{pa}(i)}, \mathbf{x}). \qquad (1)$$

### 2.1  Training

With the simplification in (1), we only need to train estimators for $p(y_i = 1 \mid y_{\mathrm{pa}(i)} = 1, \mathbf{x})$, $i \in \mathcal{T} \backslash \{0\}$. The algorithms to be proposed are independent of the way these probability estimators are learned. In the experiments, we train a multitask lasso model for each group of sibling nodes, using those training examples that their shared parent is labeled positive.

## 2.2 Prediction

For maximum a posteriori MLNP of a test pattern $\mathbf{x}$, we want to find the multilabel $\Omega^*$ that (i) maximizes the posterior probability in (1); and (ii) respects $\mathcal{T}$. Suppose that it is also known that $\mathbf{x}$ has $k$ leaf labels. The prediction task is then:

$$\Omega^* = \max_\Omega \quad p(\mathbf{y}_\Omega = \mathbf{1}, \mathbf{y}_{\Omega^c} = \mathbf{0} \mid \mathbf{x}) \tag{2}$$
$$\text{s.t.} \quad y_0 = 1, \ k \text{ of the leaves in } \mathcal{L} \text{ are labeled } 1,$$
$$\Omega \text{ contains no partial path,}$$
$$\text{all } y_i\text{'s respect the label hierarchy.} \tag{3}$$

Note that $p(\mathbf{y}_\Omega = \mathbf{1}, \mathbf{y}_{\Omega^c} = \mathbf{0} \mid \mathbf{x})$ considers all the node labels in the hierarchy simultaneously. In contrast, as discussed in Section 1, existing MLNP methods in hierarchical multiclass/multilabel classification only considers the hierarchy information locally at each node.

Associate an indicator function $\psi : \mathcal{T} \to \{0, 1\}^{N+1}$ with $\Omega$, such that $\psi_i \equiv \psi(i) = 1$ if $i \in \Omega$, and 0 otherwise. The following Proposition shows that (2) can be written as an integer linear program.

**Proposition 1.** *For a label tree, problem (2) can be rewritten as*

$$\max_\psi \quad \sum_{i \in \mathcal{T}} w_i \psi_i \tag{4}$$
$$\text{s.t.} \quad \sum_{i \in \mathcal{L}} \psi_i = k, \ \ \psi_0 = 1, \ \ \psi_i \in \{0, 1\} \ \forall i \in \mathcal{T},$$
$$\sum_{j \in child(i)} \psi_j \geq 1 \ \forall i \in \mathcal{L}^c : \psi_i = 1,$$
$$\psi_i \leq \psi_{pa(i)} \ \forall i \in \mathcal{T} \backslash \{0\}, \tag{5}$$

*where*

$$w_i = \begin{cases} \sum_{l \in child(i)} \log(1 - p_l) & i = 0 \\ \log p_i - \log(1 - p_i) & i \in \mathcal{L} \\ \log p_i - \log(1 - p_i) + \sum_{l \in child(i)} \log(1 - p_l) & i \in \mathcal{L}^c \backslash \{0\} \end{cases}, \tag{6}$$

*and* $p_i \equiv p(y_i = 1 \mid y_{pa(i)} = 1, \mathbf{x})$.

Problem (4) has $\binom{|\mathcal{L}|}{k}$ candidate solutions, which can be expensive to solve when $\mathcal{T}$ is large. In the following, we will extend the nested approximation property (NAP), first introduced in [18] for model-based compressed sensing, to constrain the optimal solution.

**Definition 1** ($k$-leaf-sparse). *A multilabel* $\mathbf{y}$ *is* $k$-leaf-sparse *if* $k$ *of the leaf nodes are labeled one.*

**Definition 2** (Nested Approximation Property (NAP)). *For a pattern* $\mathbf{x}$*, let its optimal* $k$-leaf-sparse *multilabel be* $\Omega_k$*. The NAP is satisfied if* $\{i : i \in \Omega_k\} \subset \{i : i \in \Omega_{k'}\}$ *for all* $k < k'$*.*

Note that NAP is often implicitly assumed in many HMC algorithms. For example, consider the common approach that trains a binary classifier at each node and recursively predicts from the root to the subtrees. When the classification threshold at each node is high, prediction stops early; whereas when the threshold is lowered, prediction can go further down the hierarchy. Hence, nodes that are labeled positive at a high threshold will always be labeled at a lower threshold, implying NAP. Another example is the CSSA algorithm in [11]. Since it is greedy, a larger solution (with more labels predicted positive) always includes the smaller solutions.

Algorithm 1 shows the proposed algorithm, which will be called MAS (MAndatory leaf node prediction on Structures). Similar to [11], Algorithm 1 is also greedy and based on keeping track of the *supernodes*. However, the definition of a supernode and its updating are different. Each node $i \in \mathcal{T}$ is associated with the weight $w_i$ in (6). Initially, only the root is selected ($\psi_0 = 1$). For each leaf $l$ in $\mathcal{L}$, we create a supernode, which is a subset in $\mathcal{T}$ containing all the nodes on the path from $l$ to the root. Given $|\mathcal{L}|$ leaves in $\mathcal{T}$, there are initially $|\mathcal{L}|$ supernodes. Moreover, all of them are unassigned (i.e., each contains an unselected leaf node). Each supernode $S$ has a *supernode value* (SNV) which is defined as $\text{SNV}(S) = \sum_{i \in S} w_i$.

---
**Algorithm 1** MAS (Mandatory leaf node prediction on structures).
---
1: Initialization: Initialize every node (except the root) with $\psi_i \leftarrow 0$; $\Omega \leftarrow \{0\}$; Create a supernode from each leaf with its ancestors.
2: **for** iteration=1 to $k$ **do**
3:   select the unassigned supernode $S^*$ with the largest SNV;
4:   assign all unselected nodes in $S^*$ with $\psi_i \leftarrow 1$;
5:   $\Omega \leftarrow \Omega \cup S^*$;
6:   **for** each unassigned supernode $S$ **do**
7:     update the SNV of $S$ (using Algorithm 2 for trees and Algorithm 3 for DAGs);
8:   **end for**
9: **end for**
---

In each iteration, supernode $S^*$ with the largest SNV is selected among all the unassigned supernodes. $S^*$ is then assigned, with the $\psi_i$'s of all its constituent nodes set to 1, and $\Omega$ is updated accordingly. For each remaining unassigned supernode $S$, we update its SNV to be the value that it will take if $S$ is merged with $\Omega$, i.e., $\text{SNV}(S) \leftarrow \sum_{i \in S \cup \Omega} w_i = \sum_{i \in S \setminus \Omega} w_i + \text{SNV}(\Omega)$. Since each unassigned $S$ contains exactly one leaf and we have a tree structure, this update can be implemented efficiently in $O(h^2)$ time, where $h$ is the height of the tree (Algorithm 2).

---
**Algorithm 2** Updating the SNV of an unassigned tree supernode $S$, containing the leaf $l$.
---
1: $node \leftarrow l$;
2: $\text{SNV}(S) \leftarrow \text{SNV}(\Omega)$;
3: **repeat**
4:   $\text{SNV}(S) \leftarrow \text{SNV}(S) + w_{node}$;
5:   $node \leftarrow \text{pa}(node)$;
6: **until** $node \in \Omega$.
---

---
**Algorithm 3** Updating the SNV of an unassigned DAG supernode $S$, containing the leaf $l$.
---
1: insert $l$ to $T$;
2: $\text{SNV}(S) \leftarrow \text{SNV}(\Omega)$;
3: **repeat**
4:   $node \leftarrow \text{find-max}(T)$;
5:   delete $node$ from $T$;
6:   $\text{SNV}(S) \leftarrow \text{SNV}(S) + w_{node}$;
7:   insert nodes in $\text{Pa}(node) \setminus (\Omega \cup T)$ to $T$;
8: **until** $T = \emptyset$.
---

The following Proposition shows that MAS finds the best $k$-leaf-sparse prediction.

**Proposition 2.** *Algorithm 1 obtains an optimal $\psi$ solution of (4) under the NAP assumption.*

Finally, we study the time complexity of Algorithm 1. Step 3 takes $O(|\mathcal{L}|)$ time; steps 4 and 5 take $O(h)$ time; and updating all the remaining unassigned supernodes takes $O(h^2|\mathcal{L}|)$ time. Therefore, each iteration takes $O(h^2|\mathcal{L}|)$ time, and the total time to obtain an optimal $k$-leaf-sparse solution is $O(h^2k|\mathcal{L}|)$. In contrast, a brute-force search will take $\binom{|\mathcal{L}|}{k}$ time.

### 2.2.1 Unknown Number of Labels

In practice, the value of $k$ may not be known. The straightforward approach is to run Algorithm 1 with $k = 1, \ldots, |\mathcal{L}|$, and find the $\Omega_k \in \{\Omega_1, \ldots, \Omega_{|\mathcal{L}|}\}$ that maximizes the posterior probability in (1). However, recall that $\Omega_k \subset \Omega_{k+1}$ under the NAP assumption. Hence, we can simply set $k = |\mathcal{L}|$, and $\Omega_i$ is immediately obtained as the $\Omega$ in iteration $i$. The total time complexity is $O(h^2|\mathcal{L}|^2)$. In contrast, a brute-force search takes $O(2^{|\mathcal{L}|})$ time when $k$ is unknown.

### 2.3 MLNP that Minimizes Risk

While maximizing the posterior probability minimizes the 0-1 loss, another loss function that has been popularly used in hierarchical classification is the H-loss [12]. However, along each prediction path, H-loss only penalizes the first classification mistake closest to the root. On the other hand, we are more interested in the leaf nodes in MLNP. Hence, we will adopt the symmetric loss instead, which is defined as $\ell(\Omega, \mathring{\Omega}) = |\Omega \setminus \mathring{\Omega}| + |\mathring{\Omega} \setminus \Omega|$, where $\mathring{\Omega}$ is the true multilabel for the given $\mathbf{x}$, and $\Omega$ is the prediction. However, this weights mistakes in any part of the hierarchy equally; whereas in HMC, a mistake that occurs at the higher level of the hierarchy is usually considered more crucial.

Let $I(\cdot)$ be the indicator function that returns 1 when the argument holds, 0 otherwise. We thus incorporate the hierarchy structure into $\ell(\Omega, \mathring{\Omega})$ by extending it as $\sum_i c_i I(i \in \Omega \backslash \mathring{\Omega}) + c_i I(i \in \mathring{\Omega} \backslash \Omega)$, where $c_0 = 1, c_i = c_{\mathrm{pa}(i)}/n_{\mathrm{sibl}(i)}$ as in [3], and $n_{\mathrm{sibl}(i)}$ is the number of siblings of $i$ (including $i$ itself). Finally, one can also allow different relative importance ($\alpha \geq 0$) for the false positives and negatives, and generalize $\ell(\Omega, \mathring{\Omega})$ further as

$$\ell(\Omega, \mathring{\Omega}) = \sum_i c_i^+ I(i \in \Omega \backslash \mathring{\Omega}) + c_i^- I(i \in \mathring{\Omega} \backslash \Omega), \tag{7}$$

where $c_i^+ = \frac{2c_i}{1+\alpha}$ and $c_i^- = \frac{2\alpha c_i}{1+\alpha}$.

Given a loss function $\ell(\cdot, \cdot)$, from Bayesian decision theory, the optimal multilabel $\Omega^*$ is the one that minimizes the expected loss: $\Omega^* = \arg\min_\Omega \sum_{\mathring{\Omega}} \ell(\Omega, \mathring{\Omega}) \, p(\mathbf{y}_{\mathring{\Omega}} = \mathbf{1}, \mathbf{y}_{\mathring{\Omega}^c} = \mathbf{0}|\mathbf{x})$. The proposed formulation can be easily extended for this. The following Proposition shows that it leads to a problem very similar to (4). Extension to a DAG-structured label hierarchy is analogous.

**Proposition 3.** *With a label tree and the loss function in (7), the optimal $\Omega^*$ that minimizes the expected loss can be obtained by solving (4), but with $w_i = (c_i^+ + c_i^-)p(y_i = 1|\mathbf{x}) - c_i^+$.*

## 3   Maximum a Posteriori MLNP on Label DAGs

When the label hierarchy is a DAG $\mathcal{G}$, on using the same conditional independence simplification in Section 2, we have

$$p(y_0, y_1, \ldots, y_N|\mathbf{x}) = p(y_0|\mathbf{x}) \prod_{i \in \mathcal{G} \backslash \{0\}} p(y_i \mid \mathbf{y}_{\mathrm{Pa}(i)}, \mathbf{x}), \tag{8}$$

where $\mathrm{Pa}(i)$ is the set of parents of node $i$. The prediction task involves the same optimization problem as in (2). However, there are now two interpretations on how the labels should respect the DAG in (3) [1, 11]. The first one requires that if a node is labeled positive, all its parents must also be positive. In bioinformatics, this is also called the *true path rule* that governs the DAG-structured GO taxonomy on gene functions. The alternative is that a node can be labeled positive if at least one of its parents is positive. Here, we adopt the first interpretation which is more common.

A direct maximization of $p(y_0, y_1, \ldots, y_N|\mathbf{x})$ by (8) is NP-hard [19]. Moreover, the size of each probability table $p(y_i|\mathbf{y}_{\mathrm{Pa}(i)}, \mathbf{x})$ in (8) grows exponentially with $|\mathrm{Pa}(i)|$. Hence, it can be both impractical and inaccurate when $\mathcal{G}$ is large and the sample size is limited. In the following, we assume

$$p(y_0, y_1, \ldots, y_N|\mathbf{x}) = \frac{1}{n(\mathbf{x})} p(y_0|\mathbf{x}) \prod_{i \in \mathcal{G} \backslash \{0\}} \prod_{j \in \mathrm{Pa}(i)} p(y_i \mid y_j, \mathbf{x}), \tag{9}$$

where $n(\mathbf{x})$ is a normalization term. This follows from the approach of composite likelihood (or pseudolikelihood) [20] which replaces a difficult probability density function by a set of marginal or conditional events that are easier to evaluate. In particular, (9) corresponds to the so-called *pairwise conditional likelihood* that has been used in longitudinal studies and bioinformatics [21]. Composite likelihood has been successfully used in different applications such as genetics, spatial statistics and image analysis. The connection between composite likelihood and various (flat) multilabel classification models is also recently discussed in [21]. Moreover, by using (9), the $2^{|\mathrm{Pa}(i)|}$ numbers in the probability table $p(y_i|\mathbf{y}_{\mathrm{Pa}(i)}, \mathbf{x})$ are replaced by the $|\mathrm{Pa}(i)|$ numbers in $\{p(y_i|y_j, \mathbf{x})\}_{j \in \mathrm{Pa}(i)}$, and thus the estimates obtained are much more reliable. The following Proposition shows that maximizing (9) can be reduced to a problem similar to (4).

**Proposition 4.** *With the assumption (9), problem (2) for the label DAG can be rewritten as*

$$\max_\psi \quad \sum_{i \in \mathcal{G}} w_i \psi_i \tag{10}$$

$$s.t. \quad \sum_{i \in \mathcal{L}} \psi_i = k, \; \psi_0 = 1, \; \psi_i \in \{0, 1\} \; \forall i \in \mathcal{G},$$

$$\sum_{j \in child(i)} \psi_j \geq 1 \; \forall i \in \mathcal{L}^c : \psi_i = 1,$$

$$\psi_i \leq \psi_j \; \forall j \in Pa(i), \; \forall i \in \mathcal{G} \backslash \{0\}, \tag{11}$$

$$\text{where } w_i = \begin{cases} \sum_{l \in child(0)} \log(1 - p_{l0}) & i = 0, \\ \sum_{j \in Pa(i)} (\log p_{ij} - \log(1 - p_{ij})) & i \in \mathcal{L}, \\ \sum_{j \in Pa(i)} (\log p_{ij} - \log(1 - p_{ij})) + \sum_{l \in child(i)} \log(1 - p_{li}) & i \in \mathcal{L}^c \backslash \{0\}, \end{cases}$$

$$\text{and } p_{ij} \equiv p(y_i = 1 | y_j = 1, \mathbf{x}) \text{ for } j \in Pa(i).$$

Problem (10) is similar to problem (4), except in the definition of $w_i$ and that the hierarchy constraint (11) is more general than (5). When the DAG is indeed a tree, (10) reduces to (4), and Proposition 4 reduces to Proposition 1. When $k$ is unknown, the same procedure in Section 2.2.1 applies.

In the proof of Proposition 2, we do not constrain the number of parents for each node. Hence, (10) can be solved efficiently as before, except for two modifications: (i) Each initial supernode now contains a leaf and its ancestors along all paths to the root. (ii) Since $Pa(i)$ is a set and the hierarchy is a DAG, updating the SNV gets more complicated. In Algorithm 3, $T$ is a self-balancing binary search tree (BST) that keeps track of the nodes in $S \backslash \Omega$ using their topological order[1]. To facilitate the checking of whether a node is in $\Omega$ (step 7), $\Omega$ also stores its nodes in a self-balancing BST.

Recall that for a self-balancing BST, the operations of insert, delete, find-max and finding an element all take $O(\log V)$ time, where $V \leq N$ is the number of nodes in the BST. Hence, updating the SNV of one supernode by Algorithm 3 takes $O(N \log N)$ time. As $O(|\mathcal{L}|)$ supernodes need to be updated in each iteration of Algorithm 1, this step (which is the most expensive step in Algorithm 1) takes $O(|\mathcal{L}| \cdot N \log N)$ time. The total time for Algorithm 1 is $O(k \cdot |\mathcal{L}| \cdot N \log N)$.

# 4 Experiments

In this section, experiments are performed on a number of benchmark multilabel data sets[2], with both tree- and DAG-structured label hierarchies (Table 1). As pre-processing, we remove examples that contain partial label paths and nodes with fewer than 10 positive examples. At each parent node, we then train a multitask lasso model with logistic loss using the MALSAR package [22].

## 4.1 Classification Performance

The proposed MAS algorithm is compared with HMC-LP [14], the only existing algorithm that can perform MLNP on trees (but not on DAGs). We also compare with the combined use of MetaLabeler [13] and NMLNP methods as described in Section 1. These NMLNP methods include (i) HBR, which is modified from the hierarchical classifier H-SVM [3], by replacing its base learner SVM with the multitask lasso as for MAS; (ii) CLUS-HMC [1]; and (iii) flat BR [23], which is a popular MLNP method but does not use the hierarchy information. For performance evaluation, we use the hierarchical F-measure (HF) which has been commonly used in hierarchical classification [4]. Results based on 5-fold cross-validation are shown in Table 1. As can be seen, MAS is always among the best on almost all data sets.

Next, we compare the methods using the loss in (7), where the relative importance for false positives vs negatives ($\alpha$) is set to be the ratio of the numbers of negative and positive training labels. Results are shown in Table 2. As can be seen, the risk-minimizing version (MASR) can always obtain the smallest loss. We also vary $\alpha$ in the range $\{\frac{1}{10}, \frac{1}{9}, \dots, \frac{1}{2}, 1, 2, \dots, 9, 10\}$. As can be seen from Figure 1, MASR consistently outperforms the other methods, sometimes by a significant margin.

Finally, Figure 2 illustrates some example query images and their misclassifications by MAS, MASR and BR on the caltech101 data set. As can be seen, even when MAS/MASR misclassifies the image, the hierarchy often helps to keep the prediction close to the true label.

## 4.2 Validating the NAP Assumption

In this section, we verify the validity of the NAP assumption. For each test pattern, we use brute-force search to find its best $k$-leaf-sparse prediction, and check if it includes the best $(k-1)$-leaf-sparse prediction. As brute-force search is very expensive, experiments are only performed on four

Table 1: HF values obtained by the various methods on all data sets. The best results and those that are not statistically worse (according to paired t-test with p-value less than 0.05) are in bold. HMC-LP and CLUS-HMC cannot be run on the caltech101 data, which is large and dense.

| data set | #pattern | #leaf | avg #leaf per pattern | (hierarchical) | | (with MetaLabeler) | | (flat) |
|---|---|---|---|---|---|---|---|---|
| | | | | MAS | HMC-LP | HBR | CLUS-HMC | BR |
| rcv1v2 subset1 | 4422 | 42 | 1.3 | **0.85** | 0.22 | 0.83 | 0.63 | 0.83 |
| rcv1v2 subset2 | 4485 | 43 | 1.3 | **0.85** | 0.21 | 0.84 | 0.64 | 0.84 |
| rcv1v2 subset3 | 4513 | 46 | 1.3 | **0.85** | 0.20 | 0.83 | 0.63 | 0.83 |
| rcv1v2 subset4 | 4569 | 44 | 1.3 | **0.86** | 0.21 | 0.84 | 0.64 | 0.84 |
| rcv1v2 subset5 | 4452 | 45 | 1.4 | **0.84** | 0.21 | 0.83 | 0.63 | 0.83 |
| delicious | 768 | 49 | 5.4 | 0.53 | 0.23 | 0.28 | **0.57** | 0.54 |
| enron | 1607 | 24 | 2.6 | **0.75** | 0.72 | 0.74 | 0.68 | 0.74 |
| wipo | 569 | 21 | 1 | **0.83** | 0.42 | **0.83** | 0.71 | **0.83** |
| caltech-101 | 9144 | 102 | 1 | **0.82** | - | **0.82** | - | 0.70 |
| seq (funcat) | 1115 | 36 | 1.8 | **0.26** | 0.15 | **0.25** | **0.26** | 0.23 |
| pheno (funcat) | 330 | 14 | 1.6 | **0.25** | 0.12 | **0.25** | 0.20 | **0.23** |
| struc (funcat) | 1065 | 33 | 1.8 | **0.23** | 0.03 | **0.25** | 0.21 | **0.24** |
| hom (funcat) | 1124 | 35 | 1.8 | **0.35** | 0.21 | **0.36** | 0.27 | **0.36** |
| cellcycle (funcat) | 1080 | 33 | 1.9 | **0.20** | 0.12 | **0.21** | 0.19 | **0.19** |
| church (funcat) | 1104 | 35 | 1.8 | 0.17 | 0.05 | 0.18 | **0.20** | 0.17 |
| derisi (funcat) | 995 | 33 | 1.8 | 0.18 | 0.08 | 0.18 | **0.21** | 0.18 |
| eisen (funcat) | 768 | 29 | 1.8 | **0.28** | 0.10 | **0.29** | **0.28** | 0.27 |
| gasch1 (funcat) | 1038 | 32 | 1.8 | 0.25 | 0.11 | 0.23 | **0.29** | 0.22 |
| gasch2 (funcat) | 1076 | 33 | 1.8 | **0.24** | 0.05 | 0.22 | **0.25** | **0.25** |
| spo (funcat) | 1053 | 32 | 1.8 | 0.18 | 0.10 | 0.18 | **0.23** | 0.18 |
| expr (funcat) | 1109 | 32 | 1.8 | **0.28** | 0.12 | 0.25 | 0.25 | **0.27** |
| seq (GO) | 518 | 32 | 3.6 | 0.52 | - | 0.58 | 0.59 | **0.61** |
| pheno (GO) | 227 | 19 | 3.5 | **0.57** | - | 0.53 | 0.49 | 0.55 |
| struc (GO) | 505 | 33 | 3.5 | 0.51 | - | 0.48 | **0.55** | 0.53 |
| hom (GO) | 507 | 29 | 3.2 | **0.65** | - | 0.60 | 0.59 | 0.63 |
| cellcycle (GO) | 484 | 29 | 3.1 | 0.49 | - | 0.49 | 0.51 | 0.51 |
| church (GO) | 511 | 28 | 3.2 | **0.57** | - | 0.50 | 0.53 | 0.54 |
| derisi (GO) | 492 | 31 | 3.4 | **0.56** | - | 0.49 | 0.53 | 0.54 |
| eisen (GO) | 404 | 28 | 3.4 | 0.48 | | 0.54 | **0.57** | **0.57** |
| gasch1 (GO) | 512 | 32 | 3.4 | **0.64** | - | 0.56 | 0.57 | 0.58 |
| gasch2 (GO) | 508 | 32 | 3.3 | **0.55** | - | 0.50 | 0.51 | 0.53 |
| spo (GO) | 494 | 32 | 3.3 | 0.50 | - | 0.47 | 0.49 | 0.51 |
| expr (GO) | 504 | 35 | 3.5 | 0.49 | - | 0.57 | 0.55 | **0.60** |

smaller data sets for $k = 2, \ldots, 10$. Figure 3 shows the percentage of test patterns satisfying the NAP assumption at different values of $k$. As can be seen, the NAP holds almost 100% of the time.

## 5 Conclusion

In this paper, we proposed a novel hierarchical multilabel classification (HMC) algorithm for mandatory leaf node prediction. Unlike many hierarchical multilabel/multiclass classification algorithms, it utilizes the global hierarchy information by finding the multilabel that has the largest posterior probability over all the node labels. By adopting a weak "nested approximation" assumption, which is already implicitly assumed in many HMC algorithms, we showed that this can be efficiently optimized by a simple greedy algorithm. Moreover, it can be extended to minimize the risk associated with the (hierarchically weighted) symmetric loss. Experiments performed on a number of real-world data sets demonstrate that the proposed algorithms are computationally simple and more accurate than existing HMC and flat multilabel classification methods.

## Acknowledgment

This research has been partially supported by the Research Grants Council of the Hong Kong Special Administrative Region under grant 614012.

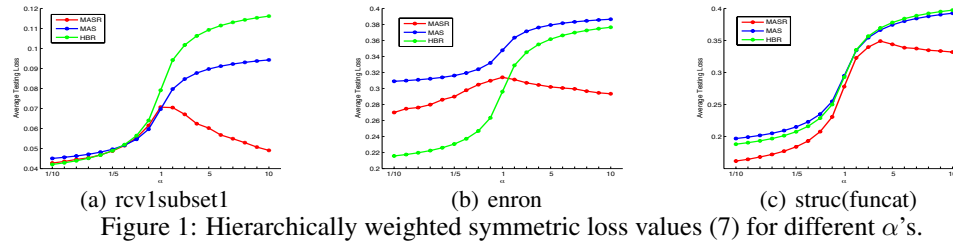

(a) rcv1subset1        (b) enron        (c) struc(funcat)

Figure 1: Hierarchically weighted symmetric loss values (7) for different $\alpha$'s.

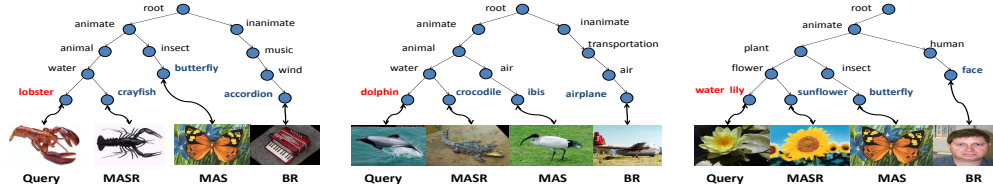

Figure 2: Example misclassifications on the caltech101 data set.

Table 2: Hierarchically weighted symmetric loss values (7) on the tree-structured data sets.

| data set | MASR | MAS | HMC-LP | (used with MetaLabeler) | | |
| | | | | HBR | CLUS-HMC | BR |
| --- | --- | --- | --- | --- | --- | --- |
| rcv1v2 subset1 | **0.05** | 0.10 | 0.46 | 0.12 | 0.20 | 0.13 |
| rcv1v2 subset2 | **0.04** | 0.09 | 0.45 | 0.11 | 0.19 | 0.12 |
| rcv1v2 subset3 | **0.04** | 0.09 | 0.45 | 0.11 | 0.20 | 0.12 |
| rcv1v2 subset4 | **0.04** | 0.10 | 0.44 | 0.11 | 0.19 | 0.11 |
| rcv1v2 subset5 | **0.05** | 0.10 | 0.46 | 0.11 | 0.20 | 0.12 |
| delicious | 0.23 | 0.19 | 0.23 | 0.14 | **0.13** | 0.14 |
| enron | 0.31 | 0.36 | **0.25** | 0.35 | 0.41 | 0.35 |
| wipo | **0.07** | 0.09 | 0.34 | 0.09 | 0.16 | 0.09 |
| caltech-101 | **0.00** | 0.01 | - | 0.01 | - | 0.01 |
| seq (funcat) | **0.24** | 0.26 | 0.41 | 0.38 | 0.38 | 0.41 |
| pheno (funcat) | **0.39** | **0.38** | 0.61 | **0.38** | 0.55 | 0.41 |
| struc (funcat) | **0.29** | 0.39 | 0.42 | 0.89 | 0.41 | 0.40 |
| hom (funcat) | **0.32** | 0.36 | 0.37 | 0.36 | 0.34 | 0.32 |
| cellcycle (funcat) | **0.24** | 0.29 | 0.41 | 0.29 | 0.38 | 0.30 |
| church (funcat) | **0.26** | 0.30 | 0.42 | 0.30 | 0.41 | 0.31 |
| derisi (funcat) | **0.26** | 0.30 | 0.45 | 0.30 | 0.43 | 0.30 |
| eisen (funcat) | **0.30** | 0.36 | 0.39 | 0.38 | 0.36 | 0.38 |
| gasch1 (funcat) | **0.24** | 0.27 | 0.43 | 0.29 | 0.39 | 0.29 |
| gasch2 (funcat) | 0.30 | **0.27** | 0.42 | 0.29 | 0.39 | 0.29 |
| spo (funcat) | 0.31 | **0.29** | 0.42 | 0.30 | 0.40 | 0.30 |
| expr (funcat) | **0.24** | 0.26 | 0.41 | 0.28 | 0.39 | 0.28 |

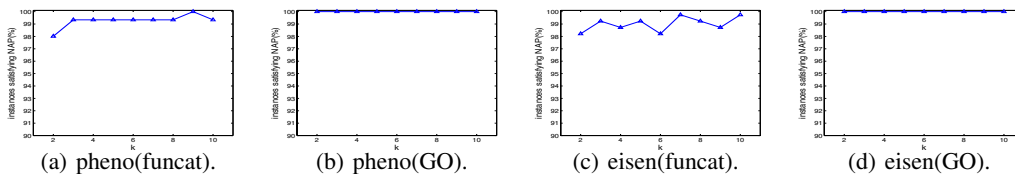

(a) pheno(funcat).      (b) pheno(GO).      (c) eisen(funcat).      (d) eisen(GO).

Figure 3: Percentage of patterns satisfying the NAP assumption at different values of $k$.

## Footnotes

[1] We number the sorted order such that nodes nearer to the root are assigned smaller values. Note that the topological sort only needs to be performed once as part of pre-processing.

[2] Downloaded from `http://mulan.sourceforge.net/datasets.html` and `http://dtai.cs.kuleuven.be/clus/hmcdatasets/`

# References

[1] C. Vens, J. Struyf, L. Schietgat, S. Dvzeroski, and H. Blockeel. Decision trees for hierarchical multi-label classification. *Machine Learning*, 73:185–214, 2008.

[2] J.J. Burred and A. Lerch. A hierarchical approach to automatic musical genre classification. In *Proceedings of the 6th International Conference on Digital Audio Effects*, 2003.

[3] N. Cesa-Bianchi, C. Gentile, and L. Zaniboni. Incremental algorithms for hierarchical classification. *Journal of Machine Learning Research*, 7:31–54, 2006.

[4] C.N. Silla and A.A. Freitas. A survey of hierarchical classification across different application domains. *Data Mining and Knowledge Discovery*, 22(1-2):31–72, 2011.

[5] Z. Barutcuoglu and O.G. Troyanskaya. Hierarchical multi-label prediction of gene function. *Bioinformatics*, 22:830–836, 2006.

[6] K. Punera, S. Rajan, and J. Ghosh. Automatically learning document taxonomies for hierarchical classification. In *Proceedings of the 14th International Conference on World Wide Web*, pages 1010–1011, 2005.

[7] M.-L. Zhang and K. Zhang. Multi-label learning by exploiting label dependency. In *Proceedings of the 16th International Conference on Knowledge Discovery and Data Mining*, pages 999–1008, 2010.

[8] S. Bengio, J. Weston, and D. Grangier. Label embedding trees for large multi-class tasks. In *Advances in Neural Information Processing Systems 23*, pages 163–171. 2010.

[9] J. Deng, S. Satheesh, A.C. Berg, and L. Fei-Fei. Fast and balanced: Efficient label tree learning for large scale object recognition. In *Advances in Neural Information Processing Systems 24*, pages 567–575. 2011.

[10] J. Rousu, C. Saunders, S. Szedmak, and J. Shawe-Taylor. Kernel-based learning of hierarchical multilabel classification models. *Journal of Machine Learning Research*, 7:1601–1626, 2006.

[11] W. Bi and J.T. Kwok. Multi-label classification on tree- and DAG-structured hierarchies. In *Proceedings of the 28th International Conference on Machine Learning*, pages 17–24, 2011.

[12] N. Cesa-Bianchi, C. Gentile, and L. Zaniboni. Hierarchical classification: Combining Bayes with SVM. In *Proceedings of the 23rd International Conference on Machine Learning*, pages 177–184, 2006.

[13] L. Tang, S. Rajan, and V.K. Narayanan. Large scale multi-label classification via metalabeler. In *Proceedings of the 18th International Conference on World Wide Web*, pages 211–220, 2009.

[14] R. Cerri, A. C. P. L. F. de Carvalho, and A. A. Freitas. Adapting non-hierarchical multilabel classification methods for hierarchical multilabel classification. *Intelligent Data Analysis*, 15:861–887, 2011.

[15] G. Tsoumakas and I. Vlahavas. Random k-labelsets: An ensemble method for multilabel classification. In *Proceedings of the 18th European Conference on Machine Learning*, pages 406–417, Warsaw, Poland, 2007.

[16] N. Cesa-Bianchi, C. Gentile, A. Tironi, and L. Zaniboni. Incremental algorithms for hierarchical classification. In *Advances in Neural Information Processing Systems 17*, pages 233–240. 2005.

[17] J.H. Zaragoza, L.E. Sucar, and EF Morales. Bayesian chain classifiers for multidimensional classification. In *Twenty-Second International Joint Conference on Artificial Intelligence*, pages 2192–2197, 2011.

[18] R.G. Baraniuk, V. Cevher, M.F. Duarte, and C. Hegde. Model-based compressive sensing. *IEEE Transactions on Information Theory*, 56:1982–2001, 2010.

[19] S.E. Shimony. Finding maps for belief networks is NP-hard. *Artificial Intelligence*, 68:399–410, 1994.

[20] C. Varin, N. Reid, and D. Firth. An overview of composite likelihood methods. *Statistica Sinica*, 21:5–42, 2011.

[21] Y. Zhang and J. Schneider. A composite likelihood view for multi-label classification. In *Proceedings of the 15th International Conference on Artificial Intelligence and Statistics*, pages 1407–1415, 2012.

[22] J. Zhou, J. Chen, and J. Ye. *MALSAR: Multi-tAsk Learning via StructurAl Regularization*. Arizona State University, 2012.

[23] G. Tsoumakas, I. Katakis, and I. Vlahavas. Mining multi-label data. In *Data Mining and Knowledge Discovery Handbook*, pages 667–685. Springer, 2010.

